# Sequential effects reflect parallel learning of multiple environmental regularities

**Matthew H. Wilder**[*]**, Matt Jones**[†]**, & Michael C. Mozer**[*]
[*]Dept. of Computer Science
[†]Dept. of Psychology
University of Colorado
Boulder, CO 80309
<wildermh@colorado.edu, mcj@colorado.edu, mozer@colorado.edu>

## Abstract

Across a wide range of cognitive tasks, recent experience influences behavior. For example, when individuals repeatedly perform a simple two-alternative forced-choice task (2AFC), response latencies vary dramatically based on the immediately preceding trial sequence. These sequential effects have been interpreted as adaptation to the statistical structure of an uncertain, changing environment (e.g., Jones and Sieck, 2003; Mozer, Kinoshita, and Shettel, 2007; Yu and Cohen, 2008). The Dynamic Belief Model (DBM) (Yu and Cohen, 2008) explains sequential effects in 2AFC tasks as a rational consequence of a dynamic internal representation that tracks second-order statistics of the trial sequence (repetition rates) and predicts whether the upcoming trial will be a repetition or an alternation of the previous trial. Experimental results suggest that first-order statistics (base rates) also influence sequential effects. We propose a model that learns both first- and second-order sequence properties, each according to the basic principles of the DBM but under a unified inferential framework. This model, the Dynamic Belief Mixture Model (DBM2), obtains precise, parsimonious fits to data. Furthermore, the model predicts dissociations in behavioral (Maloney, Martello, Sahm, and Spillmann, 2005) and electrophysiological studies (Jentzsch and Sommer, 2002), supporting the psychological and neurobiological reality of its two components.

## 1 Introduction

Picture an intense match point at the Wimbledon tennis championship, Nadal on the defense from Federer's powerful shots. Nadal returns three straight hits to his forehand side. In the split second before the ball is back in his court, he forms an expectation about where Federer will hit the ball next—will the streak of forehands continue or will there be a switch to his backhand. As the point continues, Nadal gains the upper ground and begins making Federer alternate from forehand to backhand to forehand. Now Federer finds himself trying to predict whether or not this alternating pattern will be continued with the next shot. These two are caught up in a high-stakes game of sequential effects—their actions and expectations for the current shot have a strong dependence on the past few shots. Sequential effects play a ubiquitous role in our lives—our actions are constantly affected by our recent experiences.

In controlled environments, sequential effects have been observed across a wide range of tasks and experimental paradigms, and aspects of cognition ranging from perception to memory to language to decision making. Sequential effects often occur without awareness and cannot be overridden by instructions, suggesting a robust cognitive inclination to adapt behavior in an ongoing manner. Surprisingly, people exhibit sequential effects even when they are aware that there is no dependence

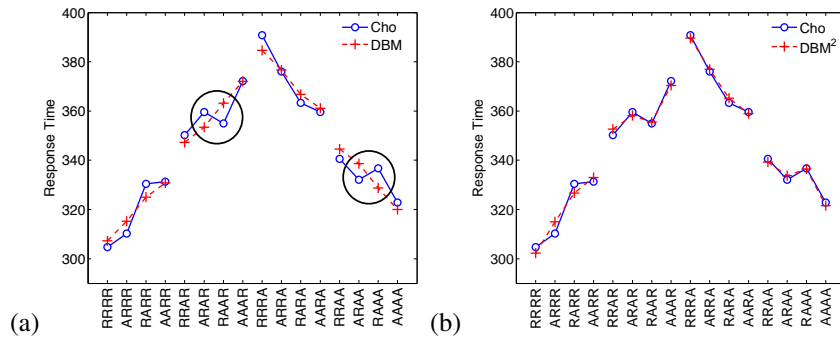

Figure 1: (a) DBM fit to the behavioral data from Cho et al. (2002). Predictions within each of the four groups are monotonically increasing or decreasing. Thus the model is unable to account for the two circled relationships. This fit accounts for 95.8% of the variance in the data. ($p_0 = \text{Beta}(2.6155, 2.4547), \alpha = 0.4899$) (b) The fit to the same data obtained from DBM2 in which probability estimates are derived from both first-order and second-order trial statistics. 99.2% of the data variance is explained by this fit. ($\alpha = 0.3427$, $w = 0.4763$)

structure to the environment. Progress toward understanding the intricate complexities of sequential effects will no doubt provide important insights into the ways in which individuals adapt to their environment and make predictions about future outcomes.

One classic domain where reliable sequential effects have been observed is in *two-alternative forced-choice* (*2AFC*) tasks (e.g, Jentzsch and Sommer, 2002; Hale, 1967; Soetens et al., 1985; Cho et al., 2002). In this type of task, participants are shown one of two different stimuli, which we denote as X and Y, and are instructed to respond as quickly as possible by mapping the stimulus to a corresponding response, say pressing the left button for X and the right button for Y. Response time (RT) is recorded, and the task is repeated several hundred or thousand times. To measure sequential effects, the RT is conditioned on the recent trial history. (In 2AFC tasks, stimuli and responses are confounded; as a result, it is common to refer to the 'trial' instead of the 'stimulus' or 'response'. In this paper, 'trial' will be synonymous with the stimulus-response pair.) Consider a sequence such as $XYYXX$, where the rightmost symbol is the current trial ($X$), and the symbols to the left are successively earlier trials. Such a four-back trial history can be represented in a manner that focuses not on the trial identities, but on whether trials are repeated or alternated. With $R$ and $A$ denoting repetitions and alternations, respectively, the trial sequence $XYYXX$ can be encoded as $ARAR$. Note that this R/A encoding collapses across isomorphic sequences $XYYXX$ and $YXXYY$.

The small blue circles in Figure 1a show the RTs from Cho et al. (2002) conditioned on the recent trial history. Along the abscissa in Figure 1a are all four-back sequence histories ordered according to the R/A encoding. The left half of the graph represents cases where the current trial is a repetition of the previous, and the right half represents cases where the current trial is an alternation. The general pattern we see in the data is a triangular shape that can be understood by comparing the two extreme points on each half, $RRRR$ vs. $AAAR$ and $RRRA$ vs. $AAAA$. It seems logical that the response to the current trial in $RRRR$ will be significantly faster than in $AAAR$ ($RT_{RRRR} < RT_{AAAR}$) because in the $RRRR$ case, the current trial matches the expectation built up over the past few trials whereas in the $AAAR$ case, the current trial violates the expectation of an alternation. The same argument applies to $RRRA$ vs. $AAAA$, leading to the intuition that $RT_{RRRA} > RT_{AAAA}$. The trial histories are ordered along the abscissa so that the left half is monotonically increasing and the right half is monotonically decreasing following the same line of intuition, i.e., many recent repetitions to many recent alternations.

## 2 Toward A Rational Model Of Sequential Effects

Many models have been proposed to capture sequential effects, including Estes (1950), Anderson (1960), Laming (1969), and Cho et al. (2002). Other models have interpreted sequential effects as adaptation to the statistical structure of a dynamic environment (e.g., Jones and Sieck, 2003; Mozer, Kinoshita, and Shettel, 2007). In this same vein, Yu and Cohen (2008) recently suggested a rational

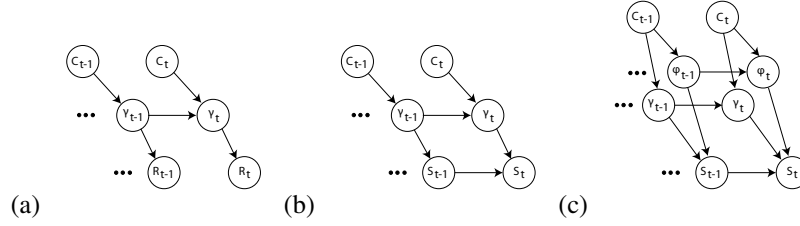

Figure 2: Three graphical models that capture sequential dependencies. (a) Dynamic Belief Model (DBM) of Yu and Cohen (2008). (b) A reformulation of DBM in which the output variable, $S_t$, is the actual stimulus identity instead of the repetition/alternation representation used in DBM. (c) Our proposed Dynamic Belief Mixture Model (DBM2). Models are explained in more detail in the text.

explanation for sequential effects such as those observed in Cho et al. (2002). According to their *Dynamic Belief Model* (*DBM*), individuals estimate the statistics of a nonstationary environment. The key contribution of this work is that it provides a rational justification for sequential effects that have been previously viewed as resulting from low-level brain mechanisms such as residual neural activation.

DBM describes performance in 2AFC tasks as Bayesian inference over whether the next trial in the sequence will be a repetition or an alternation of the previous trial, conditioned on the trial history. If $R_t$ is the Bernoulli random variable that denotes whether trial $t$ is a repetition ($R_t = 1$) or alternation ($R_t = 0$) of the previous trial, DBM determines $P(R_t|\vec{R}_{t-1})$, where $\vec{R}_{t-1}$ denotes the trial sequence preceding trial $t$, i.e., $\vec{R}_{t-1} = (R_1, R_2, ..., R_{t-1})$.

DBM assumes a generative model, shown in Figure 2a, in which $R_t = 1$ with probability $\gamma_t$ and $R_t = 0$ with probability $1 - \gamma_t$. The random variable $\gamma_t$ describes a characteristic of the environment. According to the generative model, the environment is nonstationary and $\gamma_t$ can either retain the same value as on trial $t - 1$ or it can change. Specifically, $C_t$ denotes whether the environment has changed between $t - 1$ and $t$ ($C_t = 1$) or not ($C_t = 0$). $C_t$ is a Bernoulli random variable with success probability $\alpha$. If the environment does not change, $\gamma_t = \gamma_{t-1}$. If the environment changes, $\gamma_t$ is drawn from a prior distribution, which we refer to as the *reset prior* denoted by $p_0(\gamma) \sim Beta(a, b)$.

Before each trial $t$ of a 2AFC task, DBM computes the probability of the upcoming stimulus conditioned on the trial history. The model assumes that the perceptual and motor system is tuned based on this expectation, so that RT will be a linearly decreasing function of the probability assigned to the event that actually occurs, i.e. of $P(R_t = R|\vec{R}_{t-1})$ on repetition trials and of $P(R_t = A|\vec{R}_{t-1})$ = 1 - $P(R_t = R|\vec{R}_{t-1})$ on alternation trials.

The red plusses in Figure 1 show DBM's fit to the data from Cho et al. (2002). DBM has five free parameters that were optimized to fit the data. The parameters are: the change probability, $\alpha$; the imaginary counts of the reset prior, $a$ and $b$; and two additional parameters to map model probabilities to RTs via an affine transform.

## 2.1 Intuiting DBM predictions

Another contribution of Yu and Cohen (2008) is the mathematical demonstration that DBM is approximately equivalent to an exponential filter over trial histories. That is, the probability that the current stimulus is a repetition is a weighted sum of past observations, with repetitions being scored as 1 and alternations as 0, and with weights decaying exponentially as a function of lag. The exponential filter gives insight into how DBM probabilities will vary as a function of trial history. Consider two 4-back trial histories: an alternation followed by two repetitions ($ARR-$) and two alternations followed by a repetition ($AAR-$), where the $-$ indicates that the current trial type is unknown. An exponential filter predicts that $ARR-$ will always create a stronger expectation for an R on the current trial than $AAR-$ will, because the former includes an additional past repetition. Thus, if the current trial is in fact a repetition, the model predicts a faster RT for $ARR-$ compared to $AAR-$ (i.e., $RT_{ARRR} < RT_{AARR}$). Conversely, if the current trial is an alternation, the model

predicts $RT_{ARRA} > RT_{AARA}$. Similarly, if two sequences with the same number of $R$s and $A$s are compared, for example $RAR-$ and $ARR-$, the model predicts $RT_{RARR} > RT_{ARRR}$ and $RT_{RARA} < RT_{ARRA}$ because more recent trials have a stronger influence.

Comparing the exponential filter predictions for adjacent sequences in Figure 1 yields the expectation that the RTs will be monotonically increasing in the left two groups of four and monotonically decreasing in the two right groups. The data are divided into groups of 4 because the relationships between histories like $AARR$ and $RRAR$ depend on the specific parameters of the exponential filter, which determine whether one recent $A$ will outweigh two earlier $A$s. It is clear in Figure 1 that the DBM predictions follow this pattern.

## 2.2   what's missing in DBM

DBM offers an impressive fit to the overall pattern of the behavioral data. Circled in Figure 1, however, we see two significant pairs of sequence histories for which the monotonicity prediction does not hold. These are reliable aspects of the data and are not measurement error. Consider the circle on the left, in which $RT_{ARAR} > RT_{RAAR}$ for the human data. Because DBM functions approximately as an exponential filter, and the repetition in the trial history is more recent for $ARAR$ than for $RAAR$, DBM predicts $RT_{ARAR} < RT_{RAAR}$. An exponential filter, and thus DBM, is unable to account for this deviation in the data.

To understand this mismatch, we consider an alternative representation of the trial history: the first-order sequence, i.e., the sequence of actual stimulus values. The two R/A sequences $ARAR$ and $RAAR$ correspond to stimulus sequences $XYYXX$ and $XXYXX$. If we consider an exponential filter on the actual stimulus sequence, we obtain the opposite prediction from that of DBM: $RT_{XYYXX} > RT_{XXYXX}$ because there are more recent occurrences of $X$ in the latter sequence. The other circled data in Figure 1a correspond to an analogous situation. Again, DBM also makes a prediction inconsistent with the data, that $RT_{ARAA} > RT_{RAAA}$, whereas an exponential filter on stimulus values predicts the opposite outcome—$RT_{XYYXY} < RT_{XXYXY}$. Of course this analysis leads to predictions for other pairs of points where DBM is consistent with the data and a stimulus based exponential filter is inconsistent. Nevertheless, the variations in the data suggest that more importance should be given to the actual stimulus values.

In general, we can divide the sequential effects observed in the data into two classes: first- and second-order effects. First-order sequential effects result from the priming of specific stimulus or response values. We refer to this as a first-order effect because it depends only on the stimulus values rather than a higher-order representation such as the repetition/alternation nature of a trial. These effects correspond to the estimation of the baserate of each stimulus or response value. They are observed in a wide range of experimental paradigms and are referred to as stimulus priming or response priming. The effects captured by DBM, i.e. the triangular pattern in RT data, can be thought of as a second-order effect because it reflects learning of the correlation structure between the current trial and the previous trial. In second-order effects, the actual stimulus value is irrelevant and all that matters is whether the stimulus was a repetition of the previous trial. As DBM proposes, these effects essentially arise from an attempt to estimate the repetition rate of the sequence.

DBM naturally produces second-order sequential effects because it abstracts over the stimulus level of description: observations in the model are $R$ and $A$ instead of the actual stimuli $X$ and $Y$. Because of this abstraction, DBM is inherently unable to exhibit first-order effects. To gain an understanding of how first-order effects could be integrated into this type of Bayesian framework, we reformulate the DBM architecture. Figure 2b shows an equivalent depiction of DBM in which the generative process on trial $t$ produces the actual stimulus value, denoted $S_t$. $S_t$ is conditioned on both the repetition probability, $\gamma_t$, and the previous stimulus value, $S_{t-1}$. Under this formulation, $S_t = S_{t-1}$ with probability $\gamma_t$, and $S_t$ equals the opposite of $S_{t-1}$ (i.e., $XY$ or $YX$) with probability $1 - \gamma_t$.

An additional benefit of this reformulated architecture is that it can represent first-order effects if we switch the meaning of $\gamma$. In particular, we can treat $\gamma$ as the probability of the stimulus taking on a specific value ($X$ or $Y$) instead of the probability of a repetition. $S_t$ is then simply a draw from a Bernoulli process with rate $\gamma$. Note that for modeling a first-order effect with this architecture, the conditional dependence of $S_t$ on $S_{t-1}$ becomes unnecessary. The nonstationarity of the environment, as represented by the change variable $C$, behaves in the same way regardless of whether we use the model to represent first- or second-order structure.

# 3 Dynamic Belief Mixture Model

The complex contributions of first- and second-order effects to the full pattern of observed sequential effects suggest the need for a model with more explanatory power than DBM. It seems clear that individuals are performing a more sophisticated inference about the statistics of the environment than proposed by DBM. We have shown that the DBM architecture can be reformulated to generate first-order effects by having it infer the baserate instead of the repetition rate of the sequence, but the empirical data suggest both mechanisms are present simultaneously. Thus the challenge is to merge these two effects into one model that performs joint inference over both environmental statistics.

Here we propose a Bayesian model that captures both first- and second-order effects, building on the basic principles of DBM. According to this new model, which we call the Dynamic Belief Mixture Model (DBM2), the learner assumes that the stimulus on a given trial is probabilistically affected by two factors: the random variable $\phi$, which represents the sequence baserate, and the random variable $\gamma$, which represents the repetition rate. The combination of these two factors is governed by a mixture weight $w$ that represents the relative weight of the $\phi$ component. As in DBM, the environment is assumed to be nonstationary, meaning that on each trial, with probability $\alpha$, $\phi$ and $\gamma$ are jointly resampled from the reset prior, $p_0(\phi, \gamma)$, which is uniform over $[0, 1]^2$. Figure 2c shows the graphical architecture for this model. This architecture is an extension of our reformulation of the DBM architecture in Figure 2b. Importantly, the observed variable, $S$, is the actual stimulus value instead of the repetition/alternation representation used in DBM. This architecture allows for explicit representation of the baserate, through the direct influence of $\phi_t$ on the physical stimulus value $S_t$, as well as representation of the repetition rate through the joint influence of $\gamma_t$ and the previous stimulus $S_{t-1}$ on $S_t$. Formally, we express the probability of $S_t$ given $\phi$, $\gamma$, and $S_{t-1}$ as shown in Equation 1.

$$P(S_t = X | \phi_t, \gamma_t, S_{t-1} = X) = w\phi_t + (1-w)\gamma_t$$
$$P(S_t = X | \phi_t, \gamma_t, S_{t-1} = Y) = w\phi_t + (1-w)(1-\gamma_t) \tag{1}$$

DBM2 operates by maintaining the iterative prior over $\phi$ and $\gamma$, $p(\phi_t, \gamma_t | \vec{S}_{t-1})$. After each observation, the joint posterior, $p(\phi_t, \gamma_t | \vec{S}_t)$, is computed using Bayes' Rule from the iterative prior and the likelihood of the most recent observation, as shown in Equation 2.

$$p(\phi_t, \gamma_t | \vec{S}_t) \propto P(S_t | \phi_t, \gamma_t, S_{t-1}) p(\phi_t, \gamma_t | \vec{S}_{t-1}). \tag{2}$$

The iterative prior for the next trial is then a mixture of the posterior from the current trial, weighted by $1 - \alpha$, and the reset prior, weighted by $\alpha$ (the probability of change in $\phi$ and $\gamma$).

$$p(\phi_{t+1}, \gamma_{t+1} | \vec{S}_t) = (1-\alpha)p(\phi_t, \gamma_t | \vec{S}_t) + \alpha p_0(\phi_{t+1}, \gamma_{t+1}). \tag{3}$$

The model generates predictions, $P(S_t | \vec{S}_{t-1})$, by integrating Equation 1 over the iterative prior on $\phi_t$ and $\gamma_t$. In our simulations, we maintain a discrete approximation to the continuous joint iterative prior with the interval [0,1] divided into 100 equally spaced sections. Expectations are computed by summing over the discrete probability mass function.

Figure 1b shows that DBM2 provides an excellent fit to the Cho et al. data, explaining the combination of both first- and second-order effects. To account for the overall advantage of repetition trials over alternation trials in the data, a repetition bias had to be built into the reset prior in DBM. In DBM2, the first-order component naturally introduces an advantage for repetition trials. This occurs because the estimate of $\phi_t$ is shifted toward the value of the previous stimulus, $S_{t-1}$, thus leading to a greater expectation that the same value will appear on the current trial. This fact eliminates the need for a nonuniform reset prior in DBM2. We use a uniform reset prior in all DBM2 simulations, thus allowing the model to operate with only four free parameters: $\alpha$, $w$, and the two parameters for the affine transform from model probabilities to RTs.

The nonuniform reset prior in DBM allows it to be biased either for repetition or alternation. This flexibility is important in a model, because different experiments show different biases, and the biases are difficult to predict. For example, the Jentzsch and Sommer experiment showed little

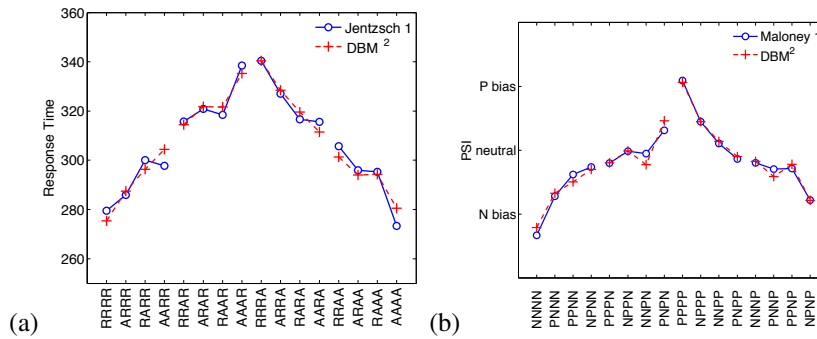

(a)                                        (b)

Figure 3: DBM2 fits for the behavioral data from (a) Jentzsch and Sommer (2002) Experiment 1 which accounts for 96.5% of the data variance ($\alpha = 0.2828$, $w = 0.3950$) and (b) Maloney et al. (2005) Experiment 1 which accounts for 97.7% of the data variance. ($\alpha = 0.0283$, $w = 0.3591$)

bias, but a replication we performed—with the same stimuli and same responses—obtained a strong alternation bias. It is our hunch that the bias should not be cast as part of the computational theory (specifically, the prior); rather, the bias reflects attentional and perceptual mechanisms at play, which can introduce varying degrees of an alternation bias. Specifically, four classic effects have been reported in the literature that make it difficult for individuals to process the same stimulus two times in a row at a short lag: attentional blink Raymond et al. (1992), inhibition of return Posner and Cohen (1984), repetition blindness Kanwisher (1987), and the Ranschburg effect Jahnke (1969). For example, with repetition blindness, processing of an item is impaired if it occurs within 500 ms of another instance of the same item in a rapid serial stream; this condition is often satisfied with 2AFC. In support of our view that fast-acting secondary mechanisms are at play in 2AFC, Jentzsch and Sommer (Experiment 2) found that using a very short lag between each response and the next stimulus modulated sequential effects in a difficult-to-interpret manner. Explaining this finding via a rational theory would be challenging. To allow for various patterns of bias across experiments, we introduced an additional parameter to our model, an offset specifically for repetition trials, which can serve as a means of removing the influence of the effects listed above. This parameter plays much the same role as DBM's priors. Although it is not as elegant, we believe it is more correct, because the bias should be considered as part of the neural implementation, not the computational theory.

## 4    Other Tests of DBM2

With its ability to represent both first- and second-order effects, DBM2 offers a robust model for a range of sequential effects. In Figure 3a, we see that DBM2 provides a close fit to the data from Experiment 1 of Jentzsch and Sommer (2002). The general design of this 2AFC task is similar to the design in Cho et al. (2002) though some details vary. Notably we see a slight advantage on alternation trials, as opposed to the repetition bias seen in Cho et al.

Surprisingly, DBM2 is able to account for the sequential effects in other binary decision tasks that do not fit into the 2AFC paradigm. In Experiment 1 of Maloney et al. (2005), subjects observed a rotation of two points on a circle and reported whether the direction of rotation was positive (clockwise) or negative (counterclockwise). The stimuli were constructed so that the direction of motion was ambiguous, but a particular variable related to the angle of motion could be manipulated to make subjects more likely to perceive one direction or the other. Psychophysical techniques were used to estimate the Point of Subjective Indifference (PSI), the angle at which the observer was equally likely to make either response. PSI measures the subject's bias toward perceiving a positive as opposed to a negative rotation. Maloney et. al. found that this bias in perceiving rotation was influenced by the recent trial history. Figure 3b shows the data for this experiment rearranged to be consistent with the R/A orderings used elsewhere (the sequences on the abscissa show the physical stimulus values, ending with Trial $t-1$). The bias, conditioned on the 4-back trial history, follows a similar pattern to that seen with RTs in Cho et al. (2002) and Jentzsch and Sommer (2002).

Table 1: A comparison between the % of data variance explained by DBM and DBM2.

|  | Cho | Jentzsch 1 | Maloney 1 |
|------|------|------------|-----------|
| DBM | 95.8 | 95.5 | 96.1 |
| DBM2 | 99.2 | 96.5 | 97.7 |

In modeling Experiment 1, we assumed that PSI reflects the subject's probabilistic expectation about the upcoming stimulus. Before each trial, we computed the model's probability that the next stimulus would be P, and then converted this probability to the PSI bias measure using an affine transform similar to our RT transform. Figure 3b shows the close fit DBM2 obtains for the experimental data.

To assess the value of DBM2, we also fit DBM to these two experiments. Table 1 shows the comparison between DBM and DBM2 for both datasets as well as Cho et al. The percentage of variance explained by the models is used as a measure for comparison. Across all three experiments, DBM2 captures a greater proportion of the variance in the data.

## 5   EEG evidence for first-order and second-order predictions

DBM2 proposes that subjects in binary choice tasks track both the baserate and the repetition rate in the sequence. Therefore an important source of support for the model would be evidence for the psychological separability of these two mechanisms. One such line of evidence comes from Jentzsch and Sommer (2002), who used electroencephalogram (EEG) recordings to provide additional insight into the mechanisms involved in the 2AFC task. The EEG was used to record subjects' lateralized readiness potential (LRP) during performance of the task. LRP essentially provides a way to identify the moment of response selection—a negative spike in the LRP signal in motor cortex reflects initiation of a response command in the corresponding hand. Jentzsch and Sommer present two different ways of analyzing the LRP data: stimulus-locked LRP (S-LRP) and response-locked LRP (LRP-R). The S-LRP interval measures the time from stimulus onset to response activation on each trial. The LRP-R interval measures the time elapsed between response activation and the actual response. Together, these two measures provide a way to divide the total RT into a stimulus-processing stage and a response-execution stage.

Interestingly, the S-LRP and LRP-R data exhibit different patterns of sequential effects when conditioned on the 4-back trial histories, as shown in Figure 4. DBM2 offers a natural explanation for the different patterns observed in the two stages of processing, because they align well with the division between first- and second-order sequential effects. In the S-LRP data, the pattern is predominantly second-order, i.e. RT on repetition trials increases as more alternations appear in the recent history, and RT on alternation trials shows the opposite dependence. In contrast, the LRP-R results exhibit an effect that is mostly first-order (which could be easily seen if the histories were reordered under an X/Y representation). Thus we can model the LRP data by extracting the separate contributions of $\phi$ and $\gamma$ in Equation 1. We use the $\gamma$ component (i.e., the second term on the RHS of Eq. 1) to predict the S-LRP results and the $\phi$ component (i.e., the first term on the RHS of Eq. 1) to predict the LRP-R results. This decomposition is consistent with the model of overall RT, because the sum of these components provides the model's RT prediction, just as the sum of the S-LRP and LRP-R measures equals the subject's actual RT (up to an additive constant explained below).

Figure 4 shows the model fits to the LRP data. The parameters of the model were constrained to be the same as those used for fitting the behavioral results shown in Figure 3a. To convert the probabilities in DBM2 to durations, we used the same scaling factor used to fit the behavioral data but allowed for new offsets for the R and A groups for both S-LRP and LRP-R. The offset terms need to be free because the difference in procedures for estimating S-LRP and LRP-R (i.e., aligning trials on the stimulus vs. the response) allows the sum of S-LRP and LRP-R to differ from total RT by an additive constant related to the random variability in RT across trials. Other than these offset terms, the fits to the LRP measures constitute parameter-free predictions of EEG data from behavioral data.

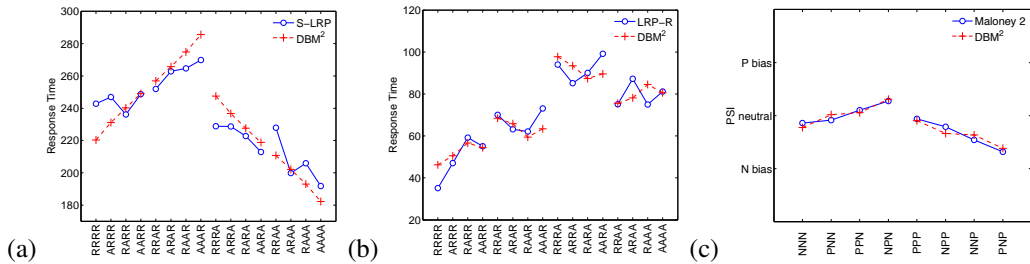

Figure 4: (a) and (b) show DBM2 fits to the S-LRP results of Jentzsch and Sommer (2002) Experiment 1. Model parameters are the same as those used for the behavioral fit shown in Figure 3a, except for offset parameters. DBM2 explains 73.4% of the variance in the S-LRP data and 87.0% of the variance in the LRP-R data. (c) Behavioral results and DBM2 fits for Experiment 2 of Maloney et al. (2005). The model fit explains 91.9% of the variance in the data ($\alpha = 0.0283$, $w = 0$).

## 6  More evidence for the two components of DBM2

In the second experiment reported in Maloney et al. (2005), participants only responded on every fourth trial. The goal of this manipulation was to test whether the sequential effect occurred in the absence of prior responses. Each ambiguous test stimulus followed three stimuli for which the direction of rotation was unambiguous and to which the subject made no response. The responses to the test stimuli were grouped according to the 3-back stimulus history, and a PSI value was computed for each of the eight histories to measure subjects' bias toward perceiving positive vs. negative rotation. The results are shown in Figure 4c. As in Figure 3b, the histories on the abscissa show the physical stimulus values, ending with Trial $t - 1$, and the arrangement of these histories is consistent with the R/A orderings used elsewhere in this paper.

DBM2's explanation of Jentzsch and Sommer's EEG results indicates that first-order sequential effects arise in response processing and second-order effects arise in stimulus processing. Therefore, the model predicts that, in the absence of prior responses, sequential effects will follow a pure second-order pattern. The results of Maloney et al.'s Experiment 2 confirm this prediction. Just as in the S-LRP data of Jentzsch and Sommer (2002), the first-order effects have mostly disappeared, and the data are well explained by a pure second-order effect (i.e., a stronger bias for alternation when there are more alternations in the history, and vice versa). We simulated this experiment with DBM2 using the same value of the change parameter ($\alpha$) from the fit of Maloney et al.'s Experiment 1. Additionally, we set the mixture parameter, $w$, to 0, which removes the first-order component of the model. For this experiment we used different affine transformation values than in Experiment 1 because the modifications in the experimental design led to a generally weaker sequential effect, which we speculate to have been due to lesser engagement by subjects when fewer responses were needed. Figure 4c shows the fit obtained by DBM2, which explains 91.9% data variance.

## 7  Discussion

Our approach highlights the power of modeling simultaneously at the levels of rational analysis and psychological mechanism. The details of the behavioral data (i.e. the systematic discrepancies from DBM) pointed to an improved rational analysis and an elaborated generative model (DBM2) that is grounded in both first- and second-order sequential statistics. In turn, the conceptual organization of the new rational model suggested a psychological architecture (i.e., separate representation of baserates and repetition rates) that was borne out in further data. The details of these latter findings now turn back to further inform the rational model. Specifically, the fits to Jentzsch and Sommer's EEG data and to Maloney et al.'s intermittent-response experiment suggest that the statistics individuals track are differentially tied to the stimuli and responses in the task. That is, rather than learning statistics of the abstract trial sequence, individuals learn the baserates (i.e., marginal probabilities) of responses and the repetition rates (i.e., transition probabilities) of stimulus sequences. This division suggests further hypotheses about both the empirical nature and the psychological representation of stimulus sequences and of response sequences, which future experiments and statistical analyses will hopefully shed light on.

# References

M. Jones and W. Sieck. Learning myopia: An adaptive recency effect in category learning. *Journal of Experimental Psychology: Learning, Memory, & Cognition*, 29:626–640, 2003.

M. Mozer, S. Kinoshita, and M. Shettel. Sequential dependencies offer insight into cognitive control. In W. Gray, editor, *Integrated Models of Cognitive Systems*, pages 180–193. Oxford University Press, 2007.

A. Yu and J. Cohen. Sequential effects: Superstition or rational behavior? *NIPS*, pages 1873–1880, 2008.

L. Maloney, M. Dal Martello, C. Sahm, and L. Spillmann. Past trials influence perception of ambiguous motion quartets through pattern completion. *Proceedings of the National Academy of Sciences*, 102:3164–3169, 2005.

I. Jentzsch and W. Sommer. Functional localization and mechanisms of sequential effects in serial reaction time tasks. *Perception and Psychophysics*, 64(7):1169–1188, 2002.

D. Hale. Sequential effects in a two-choice reaction task. *Quarterly Journal of Experimental Psychology*, 19:133–141, 1967.

E. Soetens, L. Boer, and J. Hueting. Expectancy or automatic facilitation? separating sequential effects in two-choice reaction time. *Journal of experimental psychology. Human perception and performance*, 11:598–616, 1985.

R. Cho, L. Nystrom, E. Brown, A. Jones, T. Braver, P. Holmes, and J. Cohen. Mechanisms underlying dependencies of performance on stimulus history in a two-alternative forced-choice task. *Cognitive, Affective, & Behavioral Neuroscience*, 4:283–299, 2002.

W. Estes. Toward a statistical theory of learning. *Psychological Review*, 57:94–107, 1950.

N. Anderson. Effect of first-order conditional probability in a two-choice learning situation. *Journal of Experimental Psychology*, 59(2):73–83, 1960.

D. Laming. Subjective probability in choice-reaction experiments. *Journal of Mathematical Psychology*, 6:81–120, 1969.

J. Raymond, K. Shapiro, and K. Arnell. Temporary suppression of visual processing in an rsvp task: an attentional blink? *Journal of experimental psychology. Human perception and performance*, 18:849–860, 1992.

M. Posner and Y. Cohen. Components of visual orienting. In H. Bouma and D. G. Bouwhuis, editors, *Attention and Performance X: Control of language processes*, pages 531–556. Erlbaum, Hillsdale, NJ, 1984.

N. Kanwisher. Repetition blindness: Type recognition without token individuation. *Cognition*, 27: 117–143, 1987.

J. Jahnke. The ranschburg effect. *Psychological Review*, 76:592–605, 1969.

